# Associative Memory in a Network of 'biological' Neurons

**Wulfram Gerstner** *
Department of Physics
University of California
Berkeley, CA 94720

## Abstract

The Hopfield network (Hopfield, 1982,1984) provides a simple model of an associative memory in a neuronal structure. This model, however, is based on highly artificial assumptions, especially the use of formal-two state neurons (Hopfield, 1982) or graded-response neurons (Hopfield, 1984). What happens if we replace the formal neurons by 'real' biological neurons? We address this question in two steps. First, we show that a simple model of a neuron can capture all relevant features of neuron spiking, *i.e.*, a wide range of spiking frequencies and a realistic distribution of interspike intervals. Second, we construct an associative memory by linking these neurons together. The analytical solution for a large and fully connected network shows that the Hopfield solution is valid only for neurons with a short refractory period. If the refractory period is longer than a critical duration $\gamma_c$, the solutions are qualitatively different. The associative character of the solutions, however, is preserved.

## 1 INTRODUCTION

Information received at the sensory level is encoded in spike trains which are then transmitted to different parts of the brain where the main processing steps occur. Since all the spikes of any particular neuron look alike, the information of the spike train is obviously not contained in the exact shape of the spikes, but rather in their arrival times and in the correlations between the spikes. A model neuron which tries to keep track of the voltage trace even during the spiking—like the

*present address: Physik-Department der TU Muenchen, Institut fuer Theoretische Physik,D-8046 Garching bei Muenchen

Hodgkin Huxley equations (Hodgkin, 1952) and similar models—carries therefore non-essential details, if we are only interested in the information of the spike train. On the other hand, a simple two-state neuron or threshold model is too simplistic since it cannot reproduce the variety of spiking behaviour found in real neurons. The same is true for continuous or analog model neurons which disregard the stochastic nature of neuron firing completely. In this work we construct a model of the neuron which is intermediate between these extremes. We are not concerned with the shape of the spikes and detailed voltage traces, but we want realistic interval distributions and rate functions. Finally, we link these neurons together to capture collective effects and we construct a network that can function as an associative memory.

## 2   THE MODEL NEURON

From a neural-network point of view it is often convenient to consider a neuron as a simple computational unit with no internal parameters. In this case, the neuron is described either as a 'digital' theshold unit or as a nonlinear 'analog' element with a sigmoid input-output relation. While such a simple model might be useful for formal considerations in abstract networks, it is hard to see how it could be modified to include realistic features of neurons: How can we account for the statistical properties of the spike train beyond the mean firing frequencies? What about bursting or oscillating neurons? - to mention but a few of the problems with real neurons.

We would like to use a model neuron which is closer to biology in the sense that it produces spike trains comparable of those in real neurons. Our description of the spiking dynamics therefore emphasizes three basic notions of neurobiology: *threshold, refractory period, and noise*. In particular we describe the internal state of the neuron by the membrane voltage $h$ which depends on the synaptic contributions from other neurons as well as on the spiking history of the neuron itself. In a simple threshold crossing process, a spike would be initiated as soon as the voltage $h(t)$ crosses the threshold $\theta$. Due to the statistical fluctuations of the momentary voltage around $h(t)$, however, the spiking will be a statistical event, the spikes coming a bit too early or a bit too late compared to the formal threshold crossing time, depending on the direction of the fluctuations. This fact will be taken into account by introducing a probabilistic spiking rate $r$, which depends on the difference between the membrane voltage $h$ and the threshold $\theta$ in an exponential fashion:

$$r = \frac{1}{\tau_0} \exp[\beta(h - \theta)], \tag{1}$$

where the formal temperature $\beta^{-1}$ is a measure for the noise and $\tau_0$ is an internal time constant of the neuron. If $h$ changes only slowly during a conveniently chosen time $\tau_1$, we can integrate over $\tau_1$, which yields the probability $P_F(h)$ of firing during a time step of length $\tau_1$. This gives us an analytic procedure to switch from continuous time to the discrete time step representation used later on.

If a spike is initiated in a real neuron, the neuron goes through a cycle of ion influx and efflux which changes the potential on a fast time scale and prevents immediate firing of another spike. To model this we reset the potential after each spike by

adding a negative refractory field $h^r(t)$ to the potential:

$$h(t) = h^s(t) + h^r(t), \tag{2}$$

with

$$h^r(t) = \sum_i \epsilon^r(t - t_i), \tag{3}$$

where $t_i$ is the time of the $i^{th}$ spike and $h^s(t)$ is the postsynaptic potential due to incoming spikes from other neurons. The form of the refractory function $\epsilon^r(\tau)$ together with the noise level $\beta$ determine the firing characteristics of the neuron. With fairly simple refractory fields we can achieve a sigmoid dependence of the firing frequency upon the input current (figure 1) and realistic spiking statistics (figure 3).

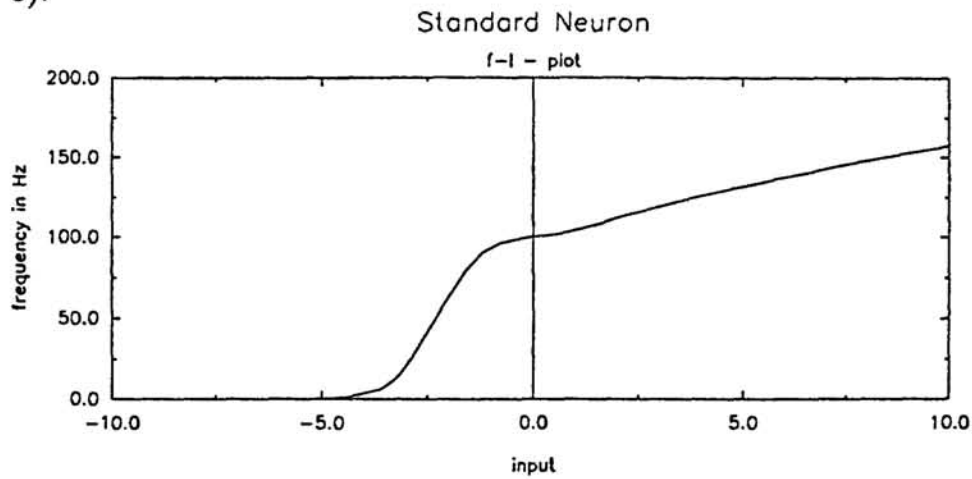

Figure 1: f-I–plot (frequency versus input current) for a standard neuron with absolute and relative refractory period. The absolute refractory period lasts for $a = 5ms$ followed by an exponentially decaying relative refractory function (time constant $2ms$). The refractory function is shown in figure 2.

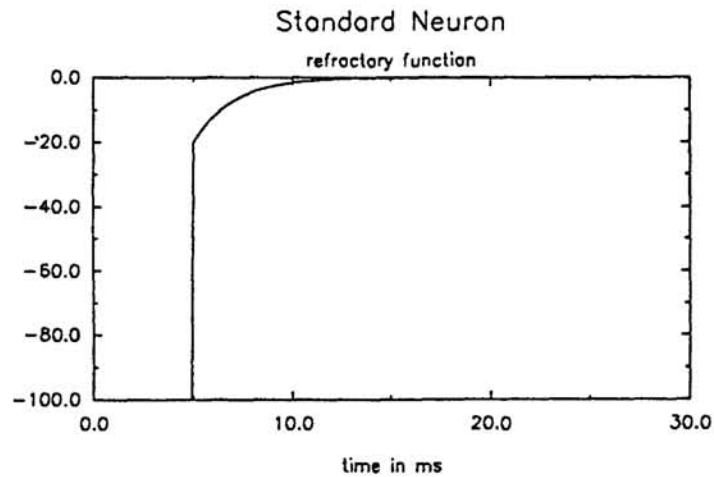

Figure 2: Refractory function of the model used in figure 1.

Indeed, the interval distribution changes from an approximate Poisson distribution for driving currents below threshold to an approximate Gaussian distribution above

threshold. Different forms of the refractory function can lead to bursting behavior or to model neurons with adaptive behavior.

In figure 4 we show a bursting neuron defined by a long-tailed refractory function with a slight overshooting at intermediate time delays. At low input level, the bursts are noise induced and appear in irregular intervals. For larger driving currents the spiking changes to regular bursting. Even a model with a simple absolute refractory period

$$\epsilon^r(\tau) = \begin{cases} -\infty & \text{if } 0 \leq \tau \leq \gamma \\ 0 & \text{otherwise} \end{cases} \tag{4}$$

has many interesting features. The explicit solution for a network of these neurons is given in the following sections.

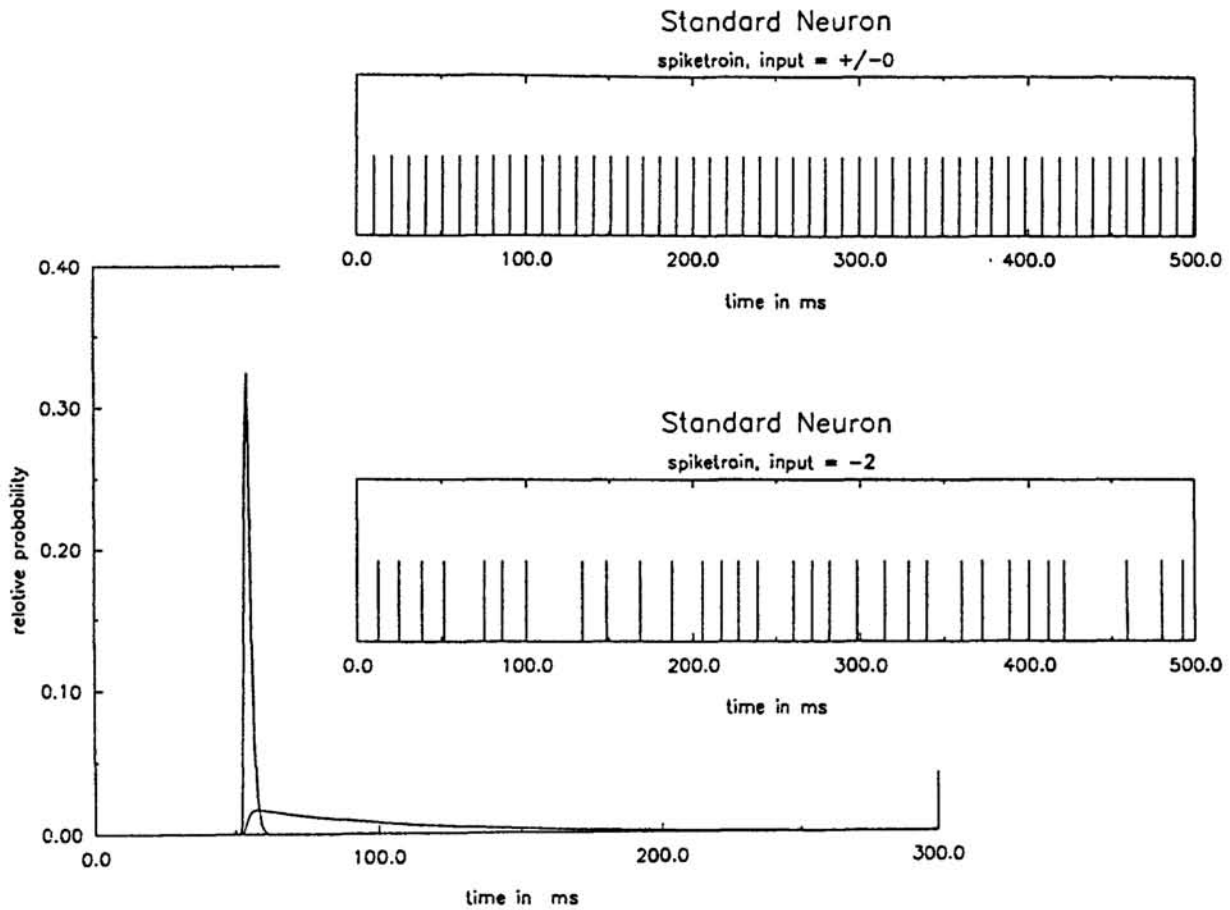

Figure 3: Spike trains and Interval distributions for the model of figure 1 at two different input levels.

## 3    THE NETWORK

So far we have only described the dynamics which initiates the spikes in the neurons. Now we have to describe the spikes themselves and their synaptic transmission to other neurons. To keep track of the spikes we assign to each neuron a two state variable $S_i$ which usually rests at $-1$ and flips to $+1$ only when a spike is initiated. In the discrete time step representation that we assume in the following the output of each neuron is then described by a sequence of Ising spins $S_i(t_n)$.

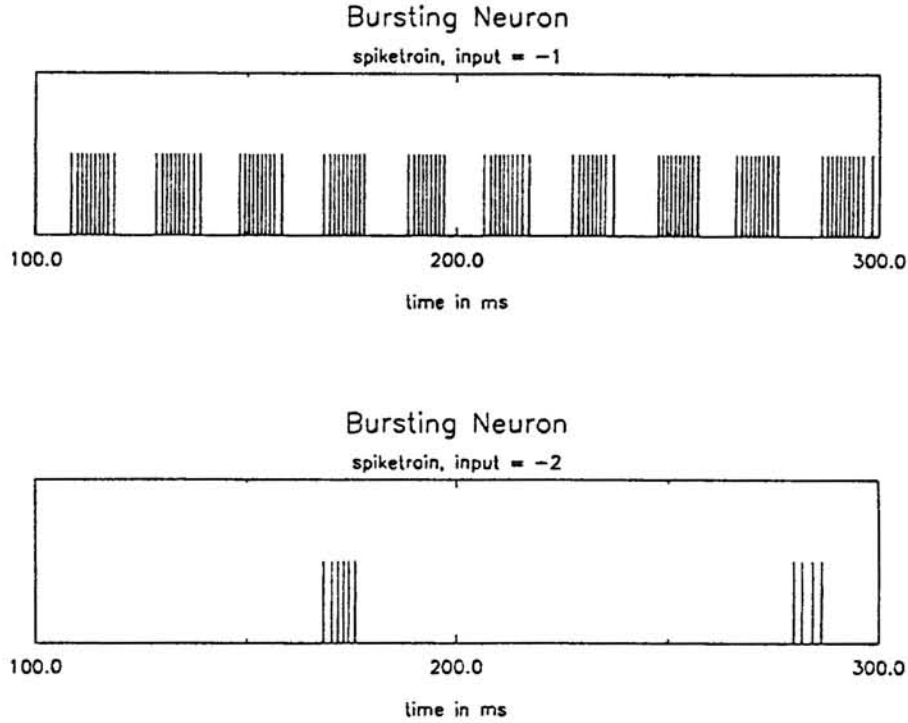

Figure 4: Spike trains for a bursting neuron. At low input level the bursts are noise induced and appear in irregular intervals, at high input level the bursting is regular.

In a network of neurons, neuron $i$ may recieve a spike from neuron $j$ via the synaptic connection, and the spike will evoke a postsynaptic potential at $i$. The strength of this response will depend on the synaptic efficacy $J_{ij}$. The time course of this response, however, can be taken to have a generic form independent of the strength of the synapse. We formalize these ideas assuming linearity and write

$$h_i^s(t_n) = \sum_j J_{ij} \sum_{\tau_m} \epsilon(\tau_m) \tilde{S}_j(t_n - \tau_m), \tag{5}$$

where $\epsilon(\tau)$ might be an experimental response function and $\tilde{S}_j$ is a conveniently normalized variable proportional to $S_j$.

For the synaptic efficacies we assume the Hebbian matrix also taken by Hopfield

$$J_{ij} = \frac{1}{N} \sum_{\mu=1}^{p} \xi_i^\mu \xi_j^\mu, \tag{6}$$

where the varables $\xi_i^\mu = \pm 1, (1 \leq i \leq N, 1 \leq \mu \leq p)$ describe the p random patterns to be stored. We can obtain these synaptic weights by a Hebbian learning procedure. It is now straightforward to incorporate the internal dynamics of the neurons, which we described in the preceding section. The refractory field can be introduced as the diagonal elements of the synaptic connection matrix

$$h_i^r(t_n) = \sum_{\tau_m} J_{ii}(\tau_m)[\tilde{S}_i(t_n - \tau_m) + 1]. \tag{7}$$

If all the neurons are equivalent, the diagonal elements must be independent of $i$ and $J_{ii}(\tau) = \epsilon^r(\tau)$ describes the generic voltage response of our model neuron after firing of a spike.

## 4   RESULTS

We can solve this model analytically in the limit of a large and fully connected network. The solution depends on an additional parameter $\rho$ which characterizes the maximum spiking frequency of the neurons. To compare our results with the Hopfield model, we replace $P_F(h)$, calculated from (1), by the generic form $\frac{1}{2}(1 + \tanh(\beta h))$ and we take the case of the simple refractory field (4). In this case the parameter $\rho$ is related to the absolute refractory period by $\rho = \frac{1}{\gamma+1}$. For a large maximum spiking frequency or $\gamma \to 0$, we recover the Hopfield solutions. For $\gamma$ larger than a critical value $\gamma_c$ the solutions are qualitatively different: there is a regime of inverse temperatures in which both the retrieval solution and the trivial solution are stable. This allows the network to remain undecided, if the initial overlap with one of the patterns is not large enough. This is in contrast to the Hopfield model (Hopfield 1982,1984) where the network is always forced into one of the retrieval states. We compared our analytic solutions with computersimulations which verified that the calculated stationary solutions are indeed stable states of the network with a wide basin of attraction. Thus the basic associative memory characteristics of the standard Hopfield model are robust under the replacement of the two state neurons by more biological neurons.

## 5   CONCLUSIONS

We constructed a network of neurons with intrinsic spiking behaviour and realistic postsynaptic response. In addition to the standard solutions we have undecided network states which might have a biological significance in the process of decision making. There remain of course a number of unbiological features in the network, *e.g.* the assumption of full connectivity, the symmetry of the connections and the linearity of the learning rule. But most of these assumptions can be overcome at least in principle (see *e.g.* Amit 1989 for references). Our results confirm the general robustness of attractor neural networks to biological modifications, but they suggest that including more biological details also adds interesting features to the variety of states available to the network.

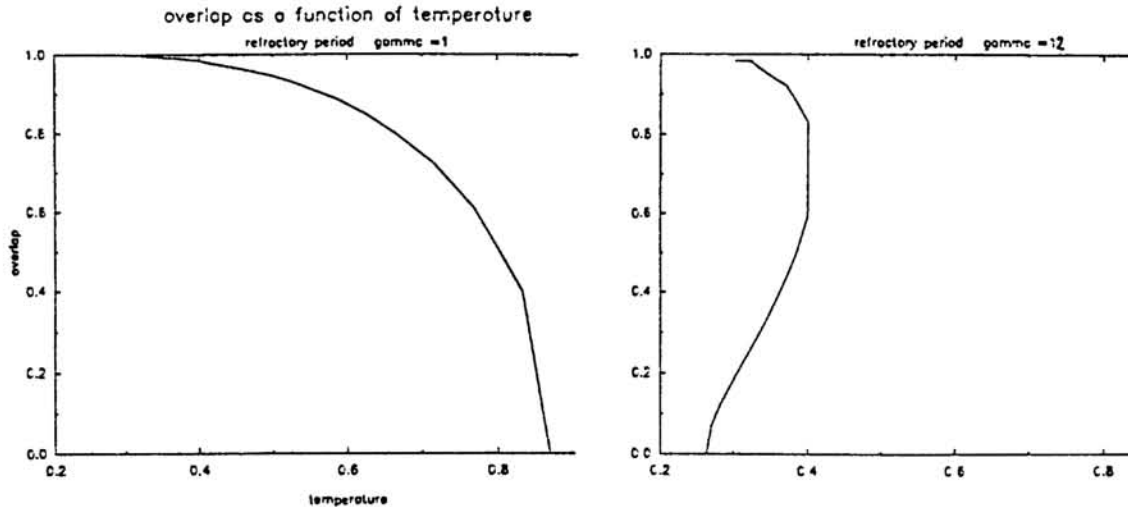

Figure 5: Stationary states of the network. Depending on the length of the refractory period the retrieval behavior varies. Figures *a* and *b* show the overlap with one of the learned patterns for different noise level $T = 1/\beta$. For a neuron a with short refractory period (figure *a*) the overlap curve is similar to those of the Hopfield model. For longer refractory periods (figure *b*) the curve is qualitatively different, showing a regime of bistability at intermediate noise levels. If the network is working at these noise levels it depends on the initial overlap with the learned pattern whether the network will go to the trivial state with overlap 0 or to the retrieval state with large overlap (overlap $m = 1$ corresponds to perfect retrieval.).

## Acknowledgements

I would like to thank William Bialek and his students at Berkeley for their generous hospitality and numerous stimulating discussions. Thanks also to J.L.vanHemmen and to Andreas Herz for many helpful comments and advice. I acknowledge the financial support of the German Academic Exchange Service (DAAD) who made my stay at Berkeley possible.

## References

Hopfield,J.J. (1982), Neural Networks and Physical Systems with Emergent Collective Computational Abilities, Proc.Natl.Acad.Sci USA **79**, 2554-2558.

Hopfield,J.J. (1984), Neurons with Graded Response have Collective Computational Properties like those of Two-State-Neurons, Proc.Natl.Acad.Sci USA **81**, 3088-3092.

Hodgkin,A.L. and Huxley,A.F. (1952) A Quantitative Description of Membrane Current and its Application to Conduction and Excitation in Nerve, J.Physiology **117**, 500-544.

Amit,D.J., (1989) Modeling Brain Function: The World of Attractor Neural Networks, CH.7. Cambridge University Press.